# MIMIC: Finding Optima by Estimating Probability Densities

**Jeremy S. De Bonet, Charles L. Isbell, Jr., Paul Viola**
Artificial Intelligence Laboratory
Massachusetts Institute of Technology
Cambridge, MA 02139

## Abstract

In many optimization problems, the structure of solutions reflects complex relationships between the different input parameters. For example, experience may tell us that certain parameters are closely related and should not be explored independently. Similarly, experience may establish that a subset of parameters must take on particular values. Any search of the cost landscape should take advantage of these relationships. We present MIMIC, a framework in which we analyze the global structure of the optimization landscape. A novel and efficient algorithm for the estimation of this structure is derived. We use knowledge of this structure to guide a randomized search through the solution space and, in turn, to refine our estimate of the structure. Our technique obtains significant speed gains over other randomized optimization procedures.

## 1 Introduction

Given some cost function $C(x)$ with local minima, we may search for the optimal $x$ in many ways. Variations of gradient descent are perhaps the most popular. When most of the minima are far from optimal, the search must either include a brute-force component or incorporate randomization. Classical examples include Simulated Annealing (SA) and Genetic Algorithms (GAs) (Kirkpatrick, Gelatt and Vecchi, 1983; Holland, 1975). In all cases, in the process of optimizing $C(x)$ many thousands or perhaps millions of samples of $C(x)$ are evaluated. Most optimization algorithms take these millions of pieces of information, and compress them into a single point $x$—the current estimate of the solution (one notable exception are GAs to which we will return shortly). Imagine splitting the search process into two parts, both taking $t/2$ time steps. Both parts are structurally identical: taking a description of $C()$, they start their search from some initial point. The sole benefit enjoyed by the second part of the search over the first is that the initial

point is perhaps closer to the optimum. Intuitively, there must be some additional information that could be learned from the first half of the search, if only to warn the second half about avoidable mistakes and pitfalls.

We present an optimization algorithm called Mutual-Information-Maximizing Input Clustering (MIMIC). It attempts to communicate information about the cost function obtained from one iteration of the search to later iterations of the search directly. It does this in an efficient and principled way. There are two main components of MIMIC: first, a randomized optimization algorithm that samples from those regions of the input space most likely to contain the minimum for $C()$; second, an effective density estimator that can be used to capture a wide variety of structure on the input space, yet is computable from simple second order statistics on the data. MIMIC's results on simple cost functions indicate an order of magnitude improvement in performance over related approaches. Further experiments on a k-color map coloring problem yield similar improvements.

## 2  Related Work

Many well known optimization procedures neither represent nor utilize the structure of the optimization landscape. In contrast, Genetic Algorithms (GA) attempt to capture this structure by an ad hoc embedding of the parameters onto a line (the chromosome). The intent of the crossover operation in standard genetic algorithms is to preserve and propagate a group of parameters that *might* be partially responsible for generating a favorable evaluation. Even when such groups exist, many of the offspring generated do not preserve the structure of these groups because the choice of crossover point is random.

In problems where the benefit of a parameter is completely independent of the value of all other parameters, the population simply encodes information about the probability distribution over each parameter. In this case, the crossover operation is equivalent to sampling from this distribution; the more crossovers the better the sample. Even in problems where fitness is obtained through the combined effects of clusters of inputs, the GA crossover operation is beneficial only when its randomly chosen clusters happen to closely match the underlying structure of the problem. Because of the rarity of such a fortuitous occurrence, the benefit of the crossover operation is greatly diminished. As as result, GAs have a checkered history in function optimization (Baum, Boneh and Garrett, 1995; Lang, 1995). One of our goals is to incorporate insights from GAs in a principled optimization framework.

There have been other attempts to capture the advantages of GAs. Population Based Incremental Learning (PBIL) attempts to incorporate the notion of a candidate population by replacing it with a single probability vector (Baluja and Caruana, 1995). Each element of the vector is the probability that a particular bit in a solution is on. During the learning process, the probability vector can be thought of as a simple model of the optimization landscape. Bits whose values are firmly established have probabilities that are close to 1 or 0. Those that are still unknown have probabilities close to 0.5.

When it is the *structure* of the components of a candidate rather than the particular values of the components that determines how it fares, it can be difficult to move PBIL's representation towards a viable solution. Nevertheless, even in these sorts of problems PBIL often out-performs genetic algorithms because those algorithms are hindered by the fact that random crossovers are infrequently beneficial.

A very distinct, but related technique was proposed by Sabes and Jordan for a

reinforcement learning task (Sabes and Jordan, 1995). In their framework, the learner must generate actions so that a reinforcement function can be completely explored. Simultaneously, the learner must exploit what it has learned so as to optimize the long-term reward. Sabes and Jordan chose to construct a Boltzmann distribution from the reinforcement function: $p(x) = \frac{\exp\left(\frac{R(x)}{T}\right)}{Z_T}$ where $R(x)$ is the reinforcement function for action $X$, $T$ is the temperature, and $Z_T$ is a normalization factor. They use this distribution to generate actions. At high temperatures this distribution approaches the uniform distribution, and results in random exploration of $R()$. At low temperatures only those actions which garner large reinforcement are generated. By reducing $T$, the learner progresses from an initially randomized search to a more directed search about the true optimal action. Interestingly, their estimate for $p(x)$ is to some extent a model of the optimization landscape which is constructed during the learning process. To our knowledge, Sabes and Jordan have neither attempted optimization over high dimensional spaces, nor attempted to fit $p(x)$ with a complex model.

## 3   MIMIC

Knowing nothing else about $C(x)$ it might not be unreasonable to search for its minimum by generating points from a uniform distribution over the inputs $p(x)$. Such a search allows none of the information generated by previous samples to effect the generation of subsequent samples. Not surprisingly, much less work might be necessary if samples were generated from a distribution, $p^\theta(x)$, that is uniformly distributed over those $x$'s where $C(x) \leq \theta$ and has a probability of 0 elsewhere. For example, if we had access to $p^{\theta_M}(x)$ for $\theta_M = \min_x C(x)$ a single sample would be sufficient to find an optimum.

This insight suggests a process of successive approximation: given a collection of points for which $C(x) \leq \theta_0$ a density estimator for $p^{\theta_0}(x)$ is constructed. From this density estimator additional samples are generated, a new threshold established, $\theta_1 = \theta_0 - \epsilon$, and a new density estimator created. The process is repeated until the values of $C(x)$ cease to improve.

The MIMIC algorithm begins by generating a random population of candidates choosen uniformly from the input space. From this population the median fitness is extracted and is denoted $\theta_0$. The algorithm then proceeds:

1. Update the parameters of the density estimator of $p^{\theta_i}(x)$ from a sample.

2. Generate more samples from the distribution $p^{\theta_i}(x)$.

3. Set $\theta_{i+1}$ equal to the Nth percentile of the data. Retain only the points less than $\theta_{i+1}$.

The validity of this approach is dependent on two critical assumptions: $p^\theta(x)$ can be successfully approximated with a finite amount of data; and $D(p^{\theta-\epsilon}(x)\|p^\theta(x))$ is small enough so that samples from $p^\theta(x)$ are also likely to be samples from $p^{\theta-\epsilon}(x)$ (where $D(p\|q)$ is the Kullback-Liebler divergence between $p$ and $q$). Bounds on these conditions can be used to prove convergence in a finite number of successive approximation steps.

The performance of this approach is dependent on the nature of the density approximator used. We have chosen to estimate the conditional distributions for every pair of parameters in the representation, a total of $O(n^2)$ numbers. In the next section we will show how we use these conditionals distributions to construct a joint distribution which is closest in the KL sense to the true joint distribution. Such an

approximator is capable of representing clusters of highly related parameters. While this might seem similar to the intuitive behavior of crossover, this representation is strictly more powerful. More importantly, our clusters are *learned* from the data, and are not pre-defined by the programmer.

## 4 Generating Events from Conditional Probabilities

The joint probability distribution over a set of random variables, $X = \{X_i\}$, is:

$$p(X) = p(X_1|X_2\ldots X_n)p(X_2|X_3\ldots X_n)\ldots p(X_{n-1}|X_n)p(X_n). \tag{1}$$

Given only pairwise conditional probabilities, $p(X_i|X_j)$ and unconditional probabilities, $p(X_i)$, we are faced with the task of generating samples that match as closely as possible the true joint distribution, $p(X)$. It is not possible to capture all possible joint distributions of $n$ variables using only the unconditional and pairwise conditional probabilities; however, we would like to describe the true joint distribution as closely as possible. Below, we derive an algorithm for choosing such a description.

Given a permutation of the numbers between 1 and $n$, $\pi = i_1 i_2 \ldots i_n$, we define a class of probability distributions, $\hat{p}_\pi(X)$:

$$\hat{p}_\pi(X) = p(X_{i_1}|X_{i_2})p(X_{i_2}|X_{i_3})\ldots p(X_{i_{n-1}}|X_{i_n})p(X_{i_n}). \tag{2}$$

The distribution $\hat{p}_\pi(X)$ uses $\pi$ as an ordering for the pairwise conditional probabilities. Our goal is to choose the permutation $\pi$ that maximizes the agreement between $\hat{p}_\pi(X)$ and the true distribution $p(X)$. The agreement between two distributions can be measured by the Kullback-Liebler divergence:

$$
\begin{aligned}
D(p||\hat{p}_\pi) &= \int_{\mathcal{X}} p[\log p - \log \hat{p}_\pi]dX \\
&= E_p[\log p] - E_p[\log \hat{p}_\pi] \\
&= -h(p) - E_p[\log p(X_{i_1}|X_{i_2})p(X_{i_2}|X_{i_3})\ldots p(X_{i_{n-1}}|X_{i_n})p(X_{i_n})] \\
&= -h(p) + h(X_{i_1}|X_{i_2}) + h(X_{i_2}|X_{i_3}) + \ldots + h(X_{i_{n-1}}|X_{i_n}) + h(X_{i_n}).
\end{aligned}
$$

This divergence is always non-negative, with equality only in the case where $\hat{p}(\pi)$ and $p(X)$ are identical distributions. The optimal $\pi$ is defined as the one that minimizes this divergence. For a distribution that can be completely described by pairwise conditional probabilities, the optimal $\pi$ will generate a distribution that will be identical to the true distribution. Insofar as the true distribution cannot be captured this way, the optimal $\hat{p}_\pi(X)$ will diverge from that distribution.

The first term in the divergence does not depend on $\pi$. Therefore, the cost function, $J_\pi(X)$, we wish to minimize is:

$$J_\pi(X) = h(X_{i_1}|X_{i_2}) + h(X_{i_2}|X_{i_3}) + \ldots + h(X_{i_{n-1}}|X_{i_n}) + h(X_{i_n}). \tag{3}$$

The optimal $\pi$ is the one that produces the lowest pairwise entropy with respect to the true distribution. By searching over all $n!$ permutations, it is possible to determine the optimal $\pi$. In the interests of computational efficiency, we employ a straightforward greedy algorithm to pick a permutation:

1. $i_n = arg\min_j \hat{h}(X_j)$.
2. $i_k = arg\min_j \hat{h}(X_j | X_{i_{k+1}})$, where
   $j \neq i_{k+1} \dots i_n$ and $k = n-1, n-2, \dots, 2, 1$.

where $\hat{h}()$ is the empirical entropy. Once a distribution is chosen, generating samples is also straightforward:

1. Choose a value for $X_{i_n}$ based on its empirical probability $\hat{p}(X_{i_n})$.
2. for $k = n-1, n-2, \dots, 2, 1$, choose element $X_{i_k}$ based on the empirical conditional probability $\hat{p}(X_{i_k} | X_{i_{k+1}})$.

The first algorithm runs in time $\mathcal{O}(n^2)$ and the second in time $\mathcal{O}(n^2)$.

## 5  Experiments

To measure the performance of MIMIC, we performed three benchmark experiments and compared our results with those obtained using several standard optimization algorithms.

We will use four algorithms in our comparisons:

1. MIMIC - the algorithm above with 200 samples per iteration
2. PBIL - standard population based incremental learning
3. RHC - randomized hill climbing
4. GA - a standard genetic algorithm with single crossover and 10% mutation rate

### 5.1  Four Peaks

The Four Peaks problem is taken from (Baluja and Caruana, 1995). Given an $N$-dimensional input vector $\vec{X}$, the four peaks evaluation function is defined as:

$$f(\vec{X}, T) = \max\left[tail(0, \vec{X}), head(1, \vec{X})\right] + R(\vec{X}, T) \tag{4}$$

where

$$tail(b, \vec{X}) = \text{number of trailing } b\text{'s in } \vec{X} \tag{5}$$

$$head(b, \vec{X}) = \text{number of leading } b\text{'s in } \vec{X} \tag{6}$$

$$R(\vec{X}, T) = \begin{cases} N & \text{if } tail(0, \vec{X}) > T \text{ and } head(1, \vec{X}) > T \\ 0 & \text{otherwise} \end{cases} \tag{7}$$

There are two global maxima for this function. They are achieved either when there are $T+1$ leading 1's followed by all 0's or when there are $T+1$ trailing 0's preceded by all 1's. There are also two suboptimal local maxima that occur with a string of all 1's or all 0's. For large values of $T$, this problem becomes increasingly more difficult because the basin of attraction for the inferior local maxima become larger.

Results for running the algorithms are shown in figure 1. In all trials, $T$ was set to be 10% of $N$, the total number of inputs. The MIMIC algorithm consistently maximizes the function with approximately one tenth the number of evaluations required by the second best algorithm.

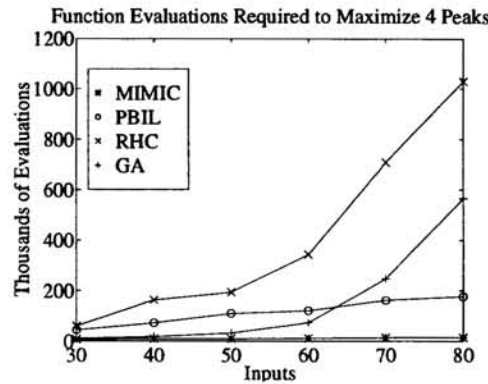

Figure 1: Number of evaluations of the Four-Peak cost function for different algorithms plotted for a variety of problems sizes.

## 5.2 Six Peaks

The Six Peaks problem is a slight variation on Four Peaks where

$$R(\vec{X}, T) = \begin{cases} N & \text{if} & tail(0, x) > T \text{ and } head(1, x) > T \text{ or} \\ & & tail(1, x) > T \text{ and } head(0, x) > T \\ 0 & & \text{otherwise} \end{cases} \tag{8}$$

This function has two additional global maxima where there are $T + 1$ leading 0's followed by all 1's or when there are $T + 1$ trailing 1's preceded by all 0's. In this case, it is not the values of the candidates that is important, but their structure: the first $T + 1$ positions should take on the same value, the last $T + 1$ positions should take on the same value, these two groups should take on different values, and the middle positions should take on all the same value.

Results for this problem are shown in figure 2. As might be expected, PBIL performed worse than on the Four Peak problem because it tends to oscillate in the middle of the space while contradictory signals pull it back and forth. The random crossover operation of the GA occasionally was able to capture some of the underlying structure, resulting in an improved relative performance of the GA. As we expected, the MIMIC algorithm was able to capture the underlying structure of the problem, and combine information from all the maxima. Thus MIMIC consistently maximizes the Six Peaks function with approximately one fiftieth the number of evaluations required by the other algorithms.

## 5.3 Max K-Coloring

A graph is K-Colorable if it is possible to assign one of $k$ colors to each of the nodes of the graph such that no adjacent nodes have the same color. Determining whether a graph is K-Colorable is known to be NP-Complete. Here, we define Max K-Coloring to be the task of finding a coloring that minimizes the number of adjacent pairs colored the same.

Results for this problem are shown in figure 2. We used a subset of graphs with a single solution (up to permutations of color) so that the optimal solution is dependent *only* on the structure of the parameters. Because of this, PBIL performs poorly. GA's perform better because *any* crossover point is representative of some of the underlying structure of the graphs used. Finally, MIMIC performs best because

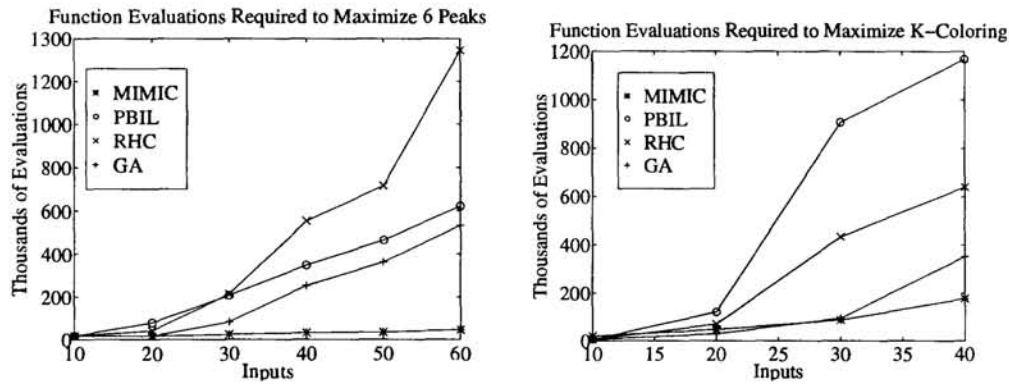

Figure 2: Number of evaluations of the Six-Peak cost function (left) and the K-Color cost function (right) for a variety of problem sizes.

it is able to capture all of the structural regularity within the inputs.

## 6   Conclusions

We have described MIMIC, a novel optimization algorithm that converges faster and more reliably than several other existing algorithms. MIMIC accomplishes this in two ways. First, it performs optimization by successively approximating the conditional distribution of the inputs given a bound on the cost function. Throughout this process, the optimum of the cost function becomes gradually more likely. As a result, MIMIC directly communicates information about the cost function from the early stages to the later stages of the search. Second, MIMIC attempts to discover common underlying structure about optima by computing second-order statistics and sampling from a distribution consistent with those statistics.

### Acknowledgments

In this research, Jeremy De Bonet is supported by the DOD Multidisciplinary Research Program of the University Research Initiative, Charles Isbell by a fellowship granted by AT&T Labs-Research, and Paul Viola by Office of Naval Research Grant No. N00014-96-1-0311. Greg Galperin helped in the preparation of this paper.

## References

Baluja, S. and Caruana, R. (1995). Removing the genetics from the standard genetic algorithm. Technical report, Carnegie Mellon Univerisity.

Baum, E. B., Boneh, D., and Garrett, C. (1995). Where genetic algorithms excel. In *Proceedings of the Conference on Computational Learning Theory*, New York. Association for Computing Machinery.

Holland, J. H. (1975). *Adaptation in Natural and Artificial Systems*. The Michigan University Press.

Kirkpatrick, S., Gelatt, C., and Vecchi, M. (1983). Optimization by Simulated Annealing. *Science*, 220(4598):671–680.

Lang, K. (1995). Hill climbing beats genetic search on a boolean circuit synthesis problem of koza's. In *Twelfth International Conference on Machine Learning*.

Sabes, P. N. and Jordan, M. I. (1995). Reinforcement learning by probability matching. In David S. Touretzky, M. M. and Perrone, M., editors, *Advances in Neural Information Processing*, volume 8, Denver 1995. MIT Press, Cambridge.